# A Model for Associative Multiplication

**G. Björn Christianson***
Department of Psychology
McMaster University
Hamilton,Ont. L8S 4K1
bjorn@caltech.edu

**Suzanna Becker**
Department of Psychology
McMaster University
Hamilton, Ont. L8S 4K1
becker@mcmaster.ca

## Abstract

Despite the fact that mental arithmetic is based on only a few hundred basic facts and some simple algorithms, humans have a difficult time mastering the subject, and even experienced individuals make mistakes. Associative multiplication, the process of doing multiplication by memory without the use of rules or algorithms, is especially problematic. Humans exhibit certain characteristic phenomena in performing associative multiplications, both in the type of error and in the error frequency. We propose a model for the process of associative multiplication, and compare its performance in both these phenomena with data from normal humans and from the model proposed by Anderson *et al* (1994).

## 1   INTRODUCTION

Associative multiplication is defined as multiplication done without recourse to computational algorithms, and as such is mainly concerned with recalling the basic times table. Learning up to the ten times table requires learning at most 121 facts; in fact, if we assume that normal humans use only four simple rules, the number of facts to be learned reduces to 39. In theory, associative multiplication is therefore a simple problem. In reality, school children find it difficult to learn, and even trained adults have a relatively high rate of error, especially in comparison to performance on associative addition, which is superficially a similar problem. There has been surprisingly little work done on the methods by which humans perform basic multiplication problems; an excellent review of the current literature is provided by McCloskey *et al* (1991).

If a model is to be considered plausible, it must have error characteristics similar to

those of humans at the same task. In arithmetic, this entails accounting for, at a minimum, two phenomena. The first is the *problem size effect*, as noted in various studies (*e.g.* Stazyk *et al*, 1982), where response times and error rates increase for problems with larger operands. Secondly, humans have a characteristic distribution in the types of errors made. Specifically, errors can be classified as one of the following five types, as suggested by Campbell and Graham (1985), Siegler (1988), McCloskey *et al* (1991), and Girelli *et al* (1996): *operand*, where the given answer is correct with one of the operands replaced (*e.g.* $4 \times 7 = 21$; this category accounts for 66.4% of all errors made by normal adults); *close-miss*, where the result is within ten percent of the correct response ($4 \times 7 = 29$; 20.0%); *table*, where the result is correct for a problem with both operands replaced ($4 \times 7 = 25$; 3.9%); *non-table*, where the result is not on the times table ($4 \times 7 = 17$; 6.7%); or *operation*, where the answer would have been correct for a different arithmetic operation, such as addition ($4 \times 7 = 11$; 3.0%)[1].

It is reasonable to assume that humans use at least two distinct representations when dealing with numbers. The work by Mandler and Shebo (1982) on modeling the performance of various species (including humans, monkeys, and pigeons) on numerosity judgment tasks suggests that in such cases a coarse coding is used. On the other hand, humans are capable of dealing with numbers as abstract symbolic concepts, suggesting the use of a precise localist coding. Previous work has either used only one of these coding ideas (for example, Sokol *et al*, 1991) or a single representation which combined aspects of both (Anderson *et al*, 1994).

Warrington (1982) documented DRC, a patient who suffered dyscalculia following a stroke. DRC retained normal intelligence and a grasp of numerical and arithmetic concepts. When presented with an arithmetic problem, DRC was capable of rapidly providing an approximate answer. However, when pressed for a precise answer, he was incapable of doing so without resorting to an explicit computational algorithm such as counting. One possible interpretation of this case study is that DRC retained the ability to work with numbers in a magnitude-related fashion, but had lost the ability to treat numbers as symbolic concepts. This suggests the hypothesis that humans may use two separate, concurrent representations for numbers: both a coarse coding and a more symbolic, precise coding in the course of doing associative arithmetic in general, and multiplication in particular, and switch between the codings at various points in the process. This hypothesis will form the basis of our modeling work. To guide the placement of these transitions between representations, we assume the further constraint that the coarse coding is the preferred coding (as it is conserved across a wide variety of species) and will tend to be expressed before the precise coding.

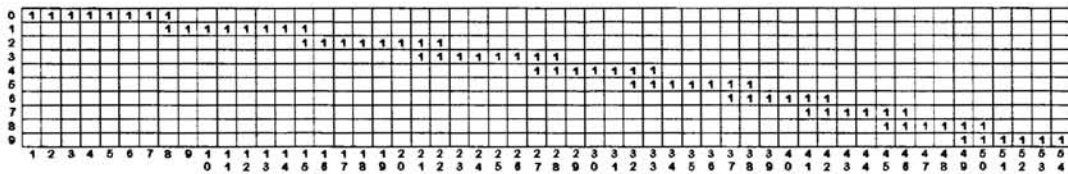

Figure 1: The coarse coding for digits. Numbers along the left are the digit; numbers along the bottom are position numbers. Blank regions in the grid represent zero activity.

## 2 METHODOLOGY

Following the work of Mandler and Shebo (1982), our coarse coding consists of a 54-dimensional vector, with a sliding "bump" of ones corresponding to the magnitude of the digit represented. The size of the bump decreases and the degree of overlap increases as the magnitude of the digit increases (Figure 1). Noise in this representation is simulated by the probability that a given bit will be in the wrong state. The precise representation, intended for symbolic manipulation of numbers, consists of a 10-dimensional vector with the value of the coded digit given by the dimension of greatest activity. Both of these representations are digit-based: each vector codes only for a number between 0 and 9, with concatenations of vectors used for numbers greater than 9.

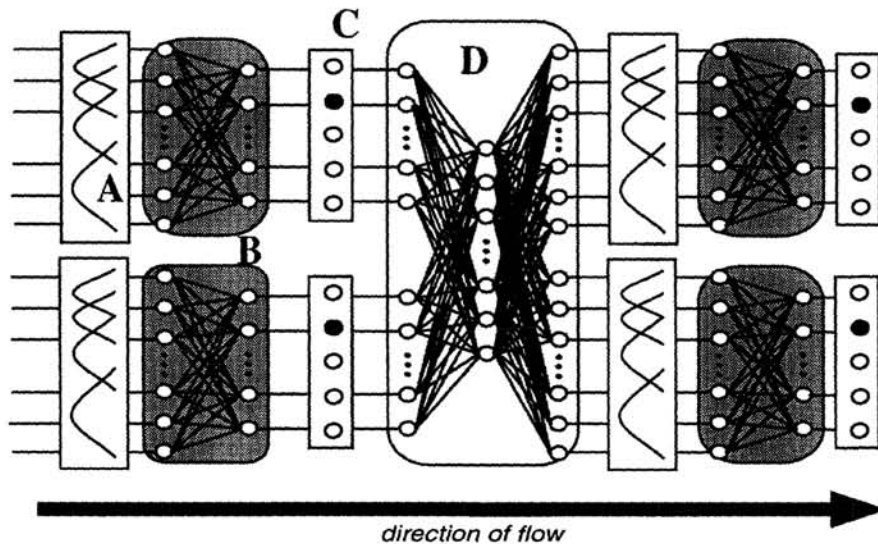

direction of flow

Figure 2: Schematic of the network architecture. (*A*) The coarse coding. (*B*) The winner-take-all network. (*C*) The precise coding. (*D*) The feed-forward look-up table. See text for details.

The model is trained in three distinct phases. A simple one-layer perceptron trained by a winner-take-all competitive learning algorithm is used to map the input operands from the original coarse coding into the precise representation. The network was trained for 10 epochs, each with a different set of 5 samples of noisy coarse-coded digits. At the end of training, the winner-take-all network performed at near-perfect levels. The translated operands are then presented to a two-layer feed-forward network with a logistic activation function trained by back-propagation. The number of hidden units was equal to the number of problems in the training set (in this case, 32) to force look-up table behaviour. The look-up table was trained independently for varying numbers of iterations, using a learning rate constant of 0.01. The output of the look-up table is coarse coded as in Figure 1. In the final phase, the table output is translated by the winner-take-all network to provide the final answer in the precise coding. A schematic of the network architecture is given in Figure 2. The operand vectors used for training of both networks had a noise parameter of 5%, while the vectors used in the analysis had 7.5% noise. Both the training and the testing problem set consisted of ten copies of each of the problems listed in Table 2, which are the problems used in

Anderson *et al* (1994). Simulations were done in MATLAB v5.1 (Mathworks, Inc., 24 Prime Park Way, Natick MA, 01760-1500).

## 3   RESULTS

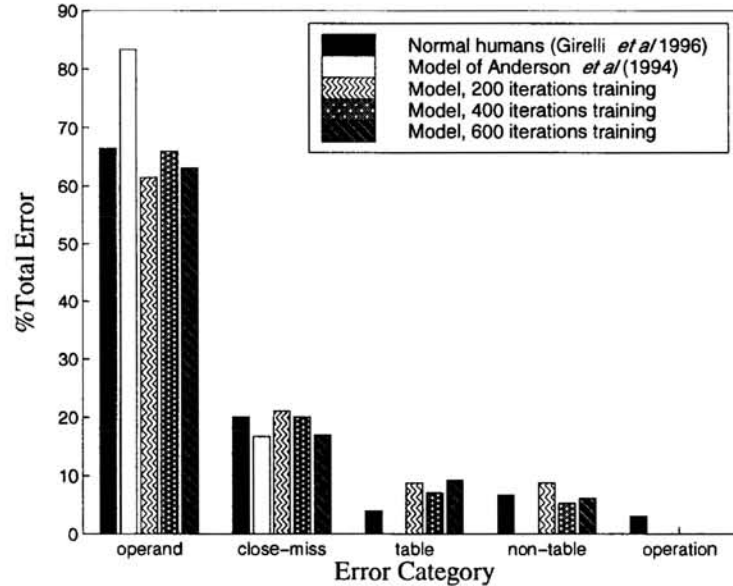

Figure 3: Error distributions for human data (Girelli *et al* 1996), the model of Anderson *et al* (1994), and our model.

Once a model has been trained, its errors on the training data can be categorized according to the error types listed in the Introduction section; a summary of the performance of our model is presented in Table 1. For comparison, we plot data generated by our model, the model of Anderson *et al* (1994), and human data from Girelli *et al* (1996) in Figure 3. In no case did the model generate an operation error. This is to be expected, as the model was only trained on multiplication, it should permit no way in which to make an operation error, other than by coincidence. A full set of results obtained from the model with 400 training iterations is presented in Table 2[2].

Table 1: Error rates generated by our model. A column for operation errors is not included, as in no instance did our model generate an operation error.

| Iterations | Errors in 320 trials | Operand (%) | Close-miss (%) | Table (%) | Non-table (%) |
|---|---|---|---|---|---|
| 200 | 114 | 61.4 | 21.0 | 8.8 | 8.8 |
| 400 | 85 | 65.9 | 20.0 | 7.1 | 7.1 |
| 600 | 65 | 63.7 | 16.9 | 9.2 | 10.8 |

[2]As in Anderson *et al* (1994), we have set $8 \times 9 = 67$ deliberately so that it is not the only problem with an answer greater than 70.

Table 2: Results from ten trials run with the model after 400 training iterations. Errors are marked in boldface.

| Problem | Trial 1 | 2 | 3 | 4 | 5 | 6 | 7 | 8 | 9 | 10 |
|---|---|---|---|---|---|---|---|---|---|---|
| 2 × 2 | 4 | 4 | 4 | 4 | 4 | 4 | 4 | 4 | 4 | 4 |
| 2 × 4 | 8 | 8 | 8 | 8 | 8 | 8 | 8 | 8 | 8 | 8 |
| 2 × 5 | 10 | 10 | 10 | 10 | 10 | 10 | 10 | 10 | 10 | 10 |
| 3 × 7 | 21 | 21 | 21 | 21 | 21 | 21 | 21 | 21 | 21 | 21 |
| 3 × 8 | 24 | 24 | 24 | **64** | 24 | 24 | **21** | 24 | 24 | **21** |
| 3 × 9 | 27 | 27 | 27 | 27 | 27 | 27 | **21** | 27 | 27 | 27 |
| 4 × 2 | 8 | 8 | 8 | 8 | 8 | 8 | 8 | **10** | 8 | 8 |
| 4 × 5 | 20 | 20 | 20 | 20 | **30** | 20 | 20 | 20 | 20 | 20 |
| 4 × 6 | 24 | 24 | 24 | **20** | **20** | 24 | 24 | **20** | 24 | **35** |
| 4 × 8 | 32 | 32 | 32 | 32 | **22** | 32 | 32 | 32 | 32 | 32 |
| 4 × 9 | 36 | 36 | 36 | 36 | **27** | 36 | 36 | **30** | 36 | 36 |
| 5 × 2 | 10 | 10 | **30** | 10 | 10 | 10 | 10 | 10 | 10 | 10 |
| 5 × 7 | **30** | **42** | **30** | 35 | 35 | 35 | **30** | **30** | 35 | 35 |
| 5 × 8 | **30** | **30** | **30** | 35 | 30 | **34** | **30** | **30** | 40 | **34** |
| 6 × 3 | **24** | 18 | 18 | **24** | **28** | **12** | 18 | 18 | **24** | **24** |
| 6 × 4 | 24 | 24 | 24 | **18** | 24 | 24 | 24 | 24 | **18** | **18** |
| 6 × 5 | 30 | 30 | 30 | 30 | 30 | 30 | 30 | 30 | 30 | 30 |
| 6 × 6 | 36 | **42** | 36 | 36 | 36 | 36 | 36 | 36 | 36 | 36 |
| 6 × 7 | 42 | **32** | **49** | 42 | 42 | 42 | 42 | 42 | 42 | 42 |
| 6 × 8 | **64** | **49** | **42** | **49** | **44** | **44** | **64** | 48 | **40** | **44** |
| 7 × 3 | **24** | 21 | 21 | 21 | 21 | 21 | 21 | 21 | 21 | **24** |
| 7 × 4 | **22** | 28 | 28 | 28 | 28 | 28 | 28 | 28 | 28 | **32** |
| 7 × 5 | 35 | 35 | 35 | 35 | 35 | **30** | 35 | 35 | 35 | 35 |
| 7 × 6 | 42 | 42 | 42 | 42 | 42 | 42 | 42 | 42 | **49** | 42 |
| 7 × 7 | **29** | 49 | 49 | 49 | 49 | **52** | 49 | **42** | 49 | **42** |
| 7 × 8 | **64** | **64** | 56 | **64** | 56 | **64** | 56 | 56 | **64** | 56 |
| 8 × 3 | 24 | 24 | **21** | 24 | **34** | 24 | 24 | 24 | 24 | 24 |
| 8 × 4 | 32 | 32 | 32 | 32 | 32 | 32 | **64** | 32 | 32 | 32 |
| 8 × 6 | **44** | **49** | **49** | **44** | **44** | **46** | **42** | **49** | **44** | **56** |
| 8 × 7 | 56 | **52** | 56 | **49** | **62** | **46** | **64** | **64** | **49** | 56 |
| 8 × 8 | 64 | 64 | 64 | 64 | **54** | 64 | 64 | 64 | 64 | 64 |
| 8 × 9 | 67 | 67 | 67 | 67 | 67 | 67 | 67 | 67 | 67 | 67 |

The convention in the current arithmetic literature is to test for the existence of a problem-size effect by fitting a line to the errors made versus the sum of operands in the problem. Positive slopes to such fits would demonstrate the existence of a problem size effect. The results of this analysis are shown in Figure 4. The model had a problem size effect in all instances. Note that no claims are made of the appropriateness of a linear model for the given data, nor should any conclusions be drawn from the specific parameters of the fit, especially given the sparsity of the data. The sole point of this analysis is to highlight a generally increasing trend.

## 4  DISCUSSION

As noted in the Results section above, our model demonstrates the problem-size effect in number of errors made (see Figure 4), though the chosen architecture does not permit a response time effect. The presence of this effect is hardly surprising, as all models which use a representation similar to our coarse coding (Mandler & Shebo, 1982; Anderson *et al*, 1994) display a problem-size effect.

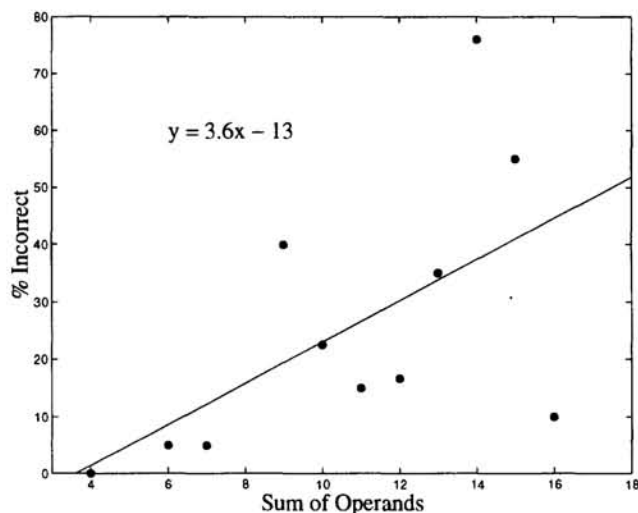

Figure 4: Demonstration of the problem size effect. The data plotted here is for the model trained for 400 iterations, as it proved the best fit to the distribution of errors in humans (Figure 3); a similar analysis gives a best-fit slope of 1.9 for 200 training iterations and 1.1 for 600 training iterations.

It has been suggested by a few researchers (*e.g.* Campbell & Graham, 1985) that the problem-size effect is simply a frequency effect, as humans encounter problems involving smaller operands more often in real life. While there is some evidence to the contrary (Hamman and Ashcraft, 1986), it remains a possibility.

It is immediately apparent from Figure 3 that our model has much the same distribution of errors as seen in normal humans, and is superior to the model of Anderson *et al* (1994) in this regard. That model, implemented as an auto-associative network using a Brain State in a Box (BSB) architecture (Anderson *et al*, 1994; Anderson 1995) generates too many operand errors, and no table, non-table or operation errors. These deficiencies can be predicted from the attractor nature of an auto-associative network. It is the process of translating between representations for digits, and the possibility for error in doing so, which we believe allows our model to produce its various categories of errors.

An interesting aspect of our model is revealed by Figure 3 and Table 1. While increased training of the look-up table improves the overall performance of the model, the error distribution remains relatively constant across the length of training studied. This suggests that in this model, the error distribution is an inherent feature of the architecture, and not a training artifact. This corresponds with data from normal humans, in which the error distribution remains relatively constant across individuals (Girelli *et al*, 1996). As noted above, the design of our model should permit the occurrence of all the various error types, save for operation errors. However, at this point, we do not have a clear understanding of the exact architectural features that generate the error distribution itself.

Defining a model for associative multiplication is only a single step towards the goal of understanding how humans perform general arithmetic. Rumelhart *et al* (1986) proposed a mechanism for multi-digit arithmetic operations given a mechanism for single-digit operations, which addresses part of the issue; this model has been implemented for addition by Cottrell and T'sung (1991). The fact that humans make operation errors suggests that there might be interactions between the mechanisms

of associative multiplication and associative addition; conversely, errors on these tasks may occur on different processing levels entirely.

In summary, this model, despite several outstanding questions, shows great potential as a description of the associative multiplication process. Eventually, we expect it to form the basis for a more complete model of arithmetic in human cognition.

## Acknowledgements

The first author acknowledges financial support from McMaster University and Industry Canada. The second author acknowledges financial support from the Natural Sciences and Engineering Research Council of Canada. We would like to thank J. Linden, D. Meeker, J. Pezaris, and M. Sahani for their feedback and comments on this work.

## Footnotes

*Author to whom correspondence should be addressed. Current address: Computation and Neural Systems, California Institute of Technology 139-74, Pasadena, CA 91125.

[1]Data taken from Girelli *et al* (1996).

## References

Anderson J.A. *et al.* (1994) In *Neural Networks for Knowledge Inference and Representation*, Levine D.S. & Aparcicio M., Eds. (Lawrence Erlbaum Associates, Hillsdale NJ) pp. 311-335.

Anderson J.A. (1995) *An Introduction to Neural Networks.* (MIT Press/Bradford, Cambridge MA) pp. 493-544.

Campbell J.I.D. & Graham D.J. (1985) *Canadian Journal of Psychology.* **39** 338.

Cottrell G.W. & T'sung F.S. (1991) In *Advances in Connectionist and Neural Computation Theory*, Burnden J.A. & Pollack J.B., Eds. (Ablex Publishing Co., Norwood NJ) pp. 305-321.

Girelli L. *et al.* (1996) *Cortex.* **32** 49.

Hamman M.S. & Ashcraft M.H. (1986) *Cognition and Instruction.* **3** 173.

Mandler G. & Shebo B.J. (1982) *Journal of Experimental Psychology: General.* **111** 1.

McCloskey M. *et al.* (1991) *Journal of Experimental Psychology: Learning, Memory, and Cognition.* **17** 377.

Rumelhart D.E. *et al.* (1986) In *Parallel distributed processing: Explorations in the microstructure of cognition. Vol. 2: Psychological and biological models*, McClelland JL, Rumelhart DE, & the PDP Research Groups, Eds. (MIT Press/Bradford, Cambridge MA) pp. 7-57.

Siegler R. (1988) *Journal of Experimental Psychology: General.* **117** 258.

Stazyk E.H. *et al.* (1982) *Journal of Experimental Psychology: Learning, Memory, and Cognition.* **8** 355.

Warrington E.K. (1982) *Quarterly Journal of Experimental Psychology.* **34A** 31.